# Accumulator networks: Suitors of local probability propagation

**Brendan J. Frey and Anitha Kannan**
Intelligent Algorithms Lab, University of Toronto, `www.cs.toronto.edu/~frey`

## Abstract

One way to approximate inference in richly-connected graphical models is to apply the sum-product algorithm (a.k.a. probability propagation algorithm), while ignoring the fact that the graph has cycles. The sum-product algorithm can be directly applied in Gaussian networks and in graphs for coding, but for many conditional probability functions – including the sigmoid function – direct application of the sum-product algorithm is not possible. We introduce "accumulator networks" that have low local complexity (but exponential global complexity) so the sum-product algorithm can be directly applied. In an accumulator network, the probability of a child given its parents is computed by accumulating the inputs from the parents in a Markov chain or more generally a tree. After giving expressions for inference and learning in accumulator networks, we give results on the "bars problem" and on the problem of extracting translated, overlapping faces from an image.

## 1 Introduction

Graphical probability models with hidden variables are capable of representing complex dependencies between variables, filling in missing data and making Bayes-optimal decisions using probabilistic inferences (Hinton and Sejnowski 1986; Pearl 1988; Neal 1992). Large, richly-connected networks with many cycles can potentially be used to model complex sources of data, such as audio signals, images and video. However, when the number of cycles in the network is large (more precisely, when the cut set size is exponential), exact inference becomes intractable. Also, to learn a probability model with hidden variables, we need to fill in the missing data using probabilistic inference, so learning also becomes intractable.

To cope with the intractability of exact inference, a variety of approximate inference methods have been invented, including Monte Carlo (Hinton and Sejnowski 1986; Neal 1992), Helmholz machines (Dayan *et al.* 1995; Hinton *et al.* 1995), and variational techniques (Jordan *et al.* 1998).

Recently, the sum-product algorithm (a.k.a. probability propagation, belief propagation) (Pearl 1988) became a major contender when it was shown to produce astounding performance on the problem of error-correcting decoding in graphs with over 1,000,000 variables and cut set sizes exceeding $2^{100,000}$ (Frey and Kschischang 1996; Frey and MacKay 1998; McEliece *et al.* 1998).

The sum-product algorithm passes messages in both directions along the edges in a graphical model and fuses these messages at each vertex to compute an estimate of $P(variable|obs)$, where *obs* is the assignment of the observed variables. In a directed

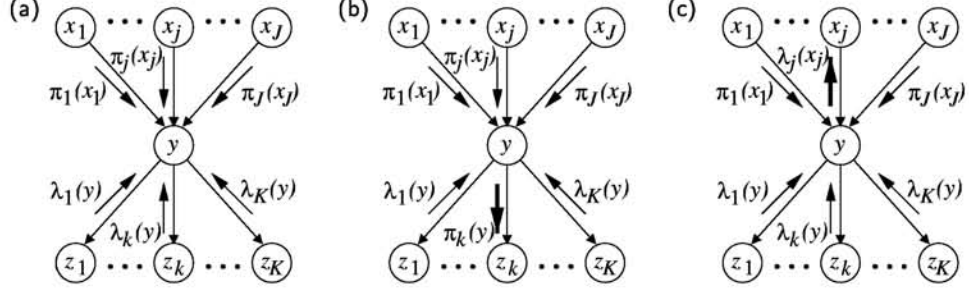

Figure 1: The sum-product algorithm passes messages in both directions along each edge in a Bayesian network. Each message is a function of the parent. (a) Incoming messages are fused to compute an estimate of $P(y|observations)$. (b) Messages are combined to produce an outgoing message $\pi_k(y)$. (c) Messages are combined to produce an outgoing message $\lambda_j(x_j)$. Initially, all messages are set to 1. Observations are accounted for as described in the text.

graphical model (Bayesian belief network) the message on an edge is a function of the parent of the edge. The messages are initialized to 1 and then the variables are processed in some order or in parallel. Each variable fuses incoming messages and produces outgoing messages, accounting for observations as described below.

If $x_1, \ldots, x_J$ are the parents of a variable $y$ and $z_1, \ldots, z_K$ are the children of $y$, messages are fused at $y$ to produce function $F(y)$ as follows (see Fig. 1a):

$$F(y) = \left(\prod_k \lambda_k(y)\right)\left(\sum_{x_1} \cdots \sum_{x_J} P(y|x_1, \ldots, x_J) \prod_j \pi_j(x_j)\right) \approx P(y, obs), \qquad (1)$$

where $P(y|x_1, \ldots, x_J)$ is the conditional probability function associated with $y$. If the graph is a tree and if messages are propagated from every variable in the network to $y$, as described below, the estimate is exact: $F(y) = P(y, obs)$. Also, normalizing $F(y)$ gives $P(y|obs)$. If the graph has cycles, this inference is approximate.

The message $\pi_k(y)$ passed from $y$ to $z_k$ is computed as follows (see Fig. 1b):

$$\pi_k(y) = F(y)/\lambda_k(y). \qquad (2)$$

The message $\lambda_j(x_j)$ passed from $y$ to $x_j$ is computed as follows (see Fig. 1c):

$$\lambda_j(x_j) = \sum_y \sum_{x_1} \cdots \sum_{x_{j-1}} \sum_{x_{j+1}} \cdots \sum_{x_J} \left(\prod_k \lambda_k(y)\right) P(y|x_1, \ldots, x_J)\left(\prod_{i \neq j} \pi_i(x_i)\right). \qquad (3)$$

Notice that $x_j$ is not summed over and is excluded from the product of the $\pi$-messages on the right.

If $y$ is observed to have the value $y^*$, the fused result at $y$ and the outgoing $\pi$ messages are modified as follows:

$$F(y) \leftarrow \begin{cases} F(y) & \text{if } y = y^* \\ 0 & \text{otherwise} \end{cases}, \quad \pi_k(y) \leftarrow \begin{cases} \pi_k(y) & \text{if } y = y^* \\ 0 & \text{otherwise} \end{cases}. \qquad (4)$$

The outgoing $\lambda$ messages are computed as follows:

$$\lambda_j(x_j) = \sum_{x_1} \cdots \sum_{x_{j-1}} \sum_{x_{j+1}} \cdots \sum_{x_J} \left(\prod_k \lambda_k(y^*)\right) P(y = y^*|x_1, \ldots, x_J)\left(\prod_{i \neq j} \pi_i(x_i)\right). \qquad (5)$$

If the graph is a tree, these formulas can be derived quite easily using the fact that summations distribute over products. If the graph is not a tree, a local independence assumption can be made to justify these formulas. In any case, the algorithm computes products and summations locally in the graph, so it is often called the "sum-product" algorithm.

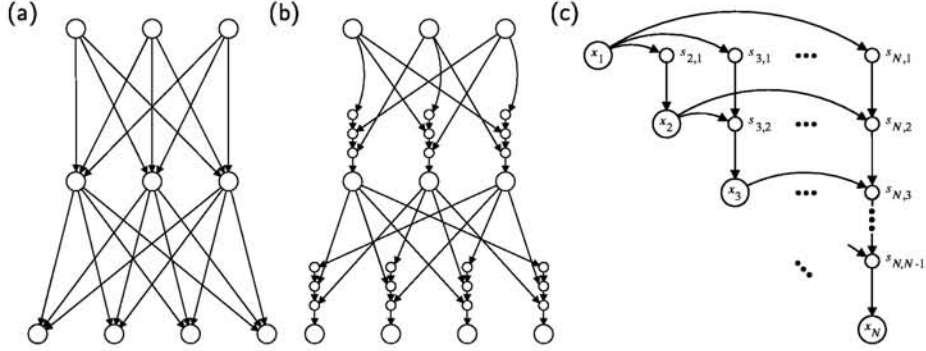

Figure 2: The local complexity of a richly connected directed graphical model such as the one in (a) can be simplified by assuming that the effects of a child's parents are accumulated by a low-complexity Markov chain as shown in (b). (c) The general structure of the "accumulator network" considered in this paper.

## 2    Accumulator networks

The complexity of the local computations at a variable generally scales exponentially with the number of parents of the variable. For example, fusion (1) requires summing over all configurations of the parents. However, for certain types of conditional probability function $P(y|x_1, \ldots, x_J)$, this exponential sum reduces to a linear-time computation. For example, if $P(y|x_1, \ldots, x_J)$ is an indicator function for $y = x_1$ XOR $x_2$ XOR $\cdots$ XOR $x_J$ (a common function for error-correcting coding), the summation can be computed in linear time using a trellis (Frey and MacKay 1998). If the variables are real-valued and $P(y|x_1, \ldots, x_J)$ is Gaussian with mean given by a linear function of $x_1, \ldots, x_J$, the integration can be computed using linear algebra (c.f. Weiss and Freeman 2000; Frey 2000).

In contrast, exact local computation for the sigmoid function, $P(y|x_1, \ldots, x_J) = 1/(1 + \exp[-\theta_0 - \sum_j \theta_j x_j])$, requires the full exponential sum. Barber (2000) considers approximating this sum using a central limit theorem approximation.

In an "accumulator network", the probability of a child given its parents is computed by accumulating the inputs from the parents in a Markov chain or more generally a tree. (For simplicity, we use Markov chains in this paper.) Fig. 2a and b show how a layered Bayesian network can be redrawn as an accumulator network. Each *accumulation variable* (state variable in the accumulation chain) has just 2 parents, and the number of computations needed for the sum-product computations for each variable in the original network now scales with the number of parents and the maximum state size of the accumulation chain in the accumulator network.

Fig. 2c shows the general form of accumulator network considered in this paper, which corresponds to a fully connected Bayes net on variables $x_1, \ldots, x_N$. In this network, the variables are $x_1, \ldots, x_N$ and the accumulation variables for $x_i$ are $s_{i,1}, \ldots, s_{i,i-1}$. The effect of variable $x_j$ on child $x_i$ is accumulated by $s_{i,j}$. The joint distribution over the variables $X = \{x_i : i = 1, \ldots, N\}$ and the accumulation variables $S = \{s_{i,j} : i = 1, \ldots, N, j = 1, \ldots, i-1\}$ is

$$P(X, S) = \prod_{i=1}^{N} \left[ \left( \prod_{j=1}^{i-1} P(s_{i,j}|x_j, s_{i,j-1}) \right) P(x_i|s_{i,i-1}) \right]. \qquad (6)$$

If $x_j$ is not a parent of $x_i$ in the original network, we set $P(s_{i,j}|x_j, s_{i,j-1}) = 1$ if $s_{i,j} = s_{i,j-1}$ and $P(s_{i,j}|x_j, s_{i,j-1}) = 0$ if $s_{i,j} \neq s_{i,j-1}$.

A well-known example of an accumulator network is the noisy-OR network (Pearl

1988; Neal 1992). In this case, all variables are binary and we set

$$P(s_{i,j} = 1 | x_j, s_{i,j-1}) = \begin{cases} 1 & \text{if } s_{i,j-1} = 1, \\ p_{i,j} & \text{if } x_j = 1 \text{ and } s_{i,j-1} = 0, \\ 0 & \text{otherwise}, \end{cases} \qquad (7)$$

where $p_{i,j}$ is the probability that $x_j = 1$ turns on the OR-chain.

Using an accumulation chain whose state space size equals the number of configurations of the parent variables, we can produce an accumulator network that can model the same joint distributions on $x_1, \ldots, x_N$ as any Bayesian network.

Inference in an accumulator network is performed by passing messages as described above, either in parallel, at random, or in a regular fashion, such as up the accumulation chains, left to the variables, right to the accumulation chains and down the accumulation chains, iteratively.

Later, we give results for an accumulator network that extracts images of translated, overlapping faces from an visual scene. The accumulation variables represent intensities of light rays at different depths in a layered 3-D scene.

## 2.1 Learning accumulator networks

To learn the conditional probability functions in an accumulator network, we apply the sum-product algorithm for each training case to compute sufficient statistics. Following Russell and Norvig (1995), the sufficient statistic needed to update the conditional probability function $P(s_{i,j} | x_j, s_{i,j-1})$ for $s_{i,j}$ in Fig. 2c is $P(s_{i,j}, x_j, s_{i,j-1} | obs)$. In particular,

$$\frac{\partial \log P(obs)}{\partial P(s_{i,j} | x_j, s_{i,j-1})} = \frac{P(s_{i,j}, x_j, s_{i,j-1} | obs)}{P(s_{i,j} | x_j, s_{i,j-1})}. \qquad (8)$$

$P(s_{i,j}, x_j, s_{i,j-1} | obs)$ is approximated by normalizing the product of $P(s_{i,j} | x_j, s_{i,j-1})$ and the $\lambda$ and $\pi$ messages arriving at $s_{i,j}$. (This approximation is exact if the graph is a tree.)

The sufficient statistics can be used for online learning or batch learning. If batch learning is used, the sufficient statistics are averaged over the training set and then the conditional probability functions are modified. In fact, the conditional probability function $P(s_{i,j} | x_j, s_{i,j-1})$ can be set equal to the normalized form of the average sufficient statistic, in which case learning performs approximate EM, where the E-step is approximated by the sum-product algorithm.

## 3  The bars problem

Fig. 3a shows the network structure for the binary bars problem and Fig. 3b shows 30 training examples. For an $N \times N$ binary image, the network has 3 layers of binary variables: 1 top-layer variable (meant to select orientation); $2N$ middle-layer variables (mean to select bars); and $N^2$ bottom-layer image variables. For large $N$, performing exact inference is computationally intractable and hence the need for approximate inference.

Accumulator networks enable efficient inference using probability propagation since local computations are made feasible. The topology of the accumulator network can be easily tailored to the bars problem, as described above.

Given an accumulator network with the proper conditional probability tables, inference computes the probability of each bar and the probability of vertical

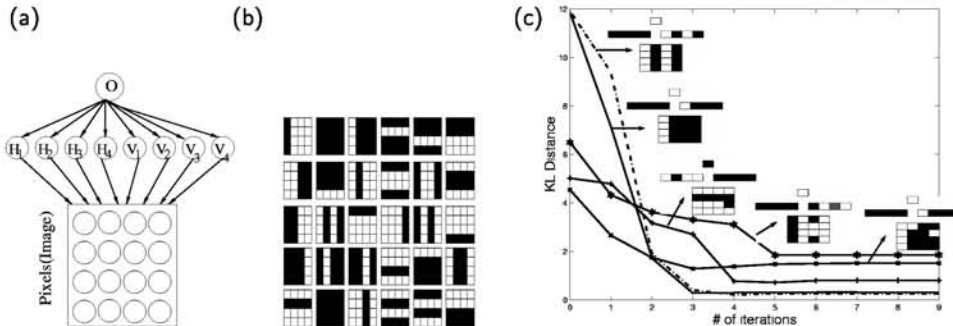

(a)          (b)          (c)

Figure 3: (a) Bayesian network for bars problem. (b) Examples of typical images. (c) KL divergence between approximate inference and exact inference after each iteration

versus horizontal orientation for an input image. After each iteration of probability propagation, messages are fused to produce estimates of these probabilities. Fig. 3c shows the Kullback Leibler divergence between these approximate probabilities and the exact probabilities after each iteration, for 5 input images. The figure also shows the most probable configuration found by approximate inference. In most cases, we found that probability propagation correctly infers the presence of appropriate bars and the overall orientation of the bars. In cases of multiple interpretations of the image (*e.g.*, Fig. 3c, image 4), probability propagation tended to find appropriate interpretations, although the divergence between the approximate and exact inferences is larger.

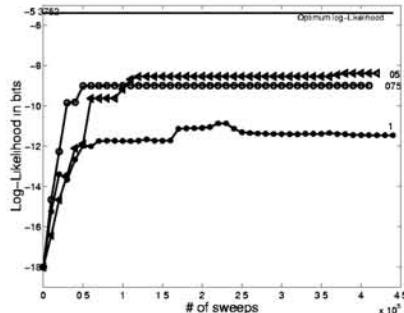

Starting with an accumulator network with random parameters, we trained the network as described above. Fig. 4 shows the online learning curves corresponding to different learning rates. The log-likelihood oscillates and although the optimum (horizontal line) is not reached, the results are encouraging.

Figure 4: Learning curves for learning rates .05, .075 and .1

## 4  Accumulating light rays for layered vision

We give results on an accumulator network that extracts image components from scenes constructed from different types of overlapping face at random positions. Suppose we divide up a 3-D scene into $L$ layers and assume that one of $O$ objects can sit in each layer in one of $P$ positions. The total number of object-position combinations per layer is $K = O \times P$. For notational convenience, we assume that each object-position pair is a different object modeled by an opaqueness map (probability that each pixel is opaque) and an appearance map (intensity of each pixel). We constrain the opaqueness and appearance maps of the same object in different positions to be the same, up to translation. Fig. 5a shows the appearance maps of 4 such objects (the first one is a wall).

In our model, $p_{kn}$ is the probability that the $n$th pixel of object $k$ is opaque and $w_{kn}$ is the intensity of the $n$th pixel for object $k$. The input images are modeled by randomly picking an object in each of $L$ layers, choosing whether each pixel in each layer is transparent or opaque, and accumulating light intensity by imaging the pixels through the layers, and then adding Gaussian noise.

Fig. 6 shows the accumulator network for this model. $z^l \in \{1, \dots, K\}$ is the index

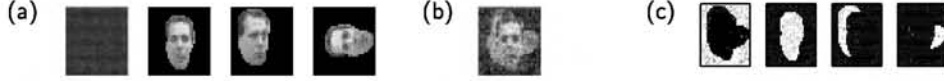

(a)    (b)    (c)

Figure 5: (a) Learned appearance maps for a wall (all pixels dark and nearly opaque) and 3 faces. (b) An image produced by combining the maps in (a) and adding noise. (c) Object-specific segmentation maps. The brightness of a pixel in the $k$th picture corresponds to the probability that the pixel is imaged by object $k$.

of the object in the $l$th layer, where layer 1 is adjacent to the camera and layer $L$ is farthest from the camera. $y_n^l$ is the accumulated discrete intensity of the light ray for pixel $n$ at layer $l$. $y_n^l$ depends on the identity of the object in the current layer $z^l$ and the intensity of pixel $n$ in the previous layer $y_n^{l+1}$. So,

$$
P(y_n^l | z^l, y_n^{l+1}) = \begin{cases} 1 & z^l = 0,\ y_n^l = y_n^{l+1} \\ 1 & z^l > 0,\ y_n^l = w_{z^l n} = y_n^{l+1} \\ p_{z^l n} & z^l > 0,\ y_n^l = w_{z^l n} \neq y_n^{l+1} \\ 1-p_{z^l n} & z^l > 0,\ y_n^l = y_n^{l+1} \neq w_{z^l n} \\ 0 & \text{otherwise.} \end{cases} \tag{9}
$$

Each condition corresponds to a different imaging operation at layer $l$ for the light ray corresponding to pixel $n$. $x_n$ is the discretized intensity of pixel $n$, obtained from the light ray arriving at the camera, $y_n^1$. $P(x_n|y_n^1)$ adds Gaussian noise to $y_n^1$.

After training the network on 200 labeled images, we applied iterative inference to identify and locate image components. After each iteration, the message passed from $y_n^l$ to $z^l$ is an estimate of the probability that the light ray for pixel $n$ is imaged by object $z^l$ at layer $l$ (*i.e.*, not occluded by other objects). So, for each object at each layer, we have an $n$-pixel "probabilistic segmentation map". In Fig. 5c we show the 4 maps in layer 1 corresponding to the objects shown in Fig. 5a, obtained after 12 iterations of the sum-product algorithm.

One such set of segmentation maps can be drawn for each layer. For deeper layers, the maps hopefully segment the part of the scene that sits behind the objects in the shallower layers. Fig. 7a shows the sets of segmentation maps corresponding to different layers, after each iteration of probability propagation, for the input image shown on the far right. After 1 iteration, the segmentation in the first layer is quite poor, causing uncertain segmen-

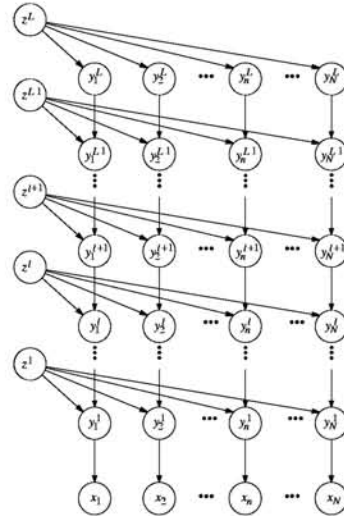

Figure 6: An accumulator network for layered vision.

tation in deeper layers (except for the wall, which is mostly segmented properly in layer 2). As iterations increases, the algorithm converges to the correct segmentation, where object 2 is in front, followed by objects 3, 4 and 1 (the wall).

It may appear from the input image in Fig. 7a that another possible depth ordering is object 2 in front, followed by objects 4, 3 and 1 – *i.e.*, objects 3 and 4 may be reversed. However, it turns out that if this were the order, a small amount of dark hair from the top of the horizontal head would be showing.

We added an extremely large amount of noise the the image used above, to see what the algorithm would do when the two depth orders really are equally likely. Fig. 7b shows the noisy image and the series of segmentation maps produced at each layer

as the number of iterations increases. The segmentation maps for layer 1 show that object 2 is correctly identified as being in the front.

Quite surprisingly, the segmentation maps in layer 2 oscillate between the two plausible interpretations of the scene – object 3 in front of object 4 and object 4 in front of object 3. Although we do not yet know how robust these oscillations are, or how accurately they reflect the probability masses in the different modes, this behavior is potentially very useful.

## References

D. Barber 2000. Tractable belief propagation. *The Learning Workshop*, Snowbird, UT.

B. J. Frey and F. R. Kschischang 1996. Probability propagation and iterative decoding. *Proceedings of the 34th Allerton Conference on Communication, Control and Computing 1996*, University of Illinois at Urbana.

B. J. Frey and D. J. C. MacKay 1998. A revolution: Belief propagation in graphs with cycles. In M. I. Jordan, M. I. Kearns and S. A. Solla (eds) *Advances in Neural Information Processing Systems 10*, MIT Press, Cambridge MA.

M. I. Jordan, Z. Ghahramani, T. S. Jaakkola and L. K. Saul 1999. An introduction to variational methods for graphical models. In M. I. Jordan (ed) *Learning in Graphical Models*, MIT Press, Cambridge, MA.

R. McEliece, D. J. C. MacKay and J. Cheng 1998. Turbodecoding as an instance of Pearl's belief propagation algorithm. *IEEE Journal on Selected Areas in Communications* **16:2**.

K. P. Murphy, Y. Weiss and M. I. Jordan 1999. Loopy belief propagation for approximate inference: An empirical study. *Proceedings of the Fifteenth Conference on Uncertainty in Artificial Intelligence*, Morgan Kaufmann, San Francisco, CA.

J. Pearl 1988. *Probabilistic Reasoning in Intelligent Systems*. Morgan Kaufmann, San Mateo CA.

S. Russell and P. Norvig 1995. *Artificial Intelligence: A Modern Approach*. Prentice-Hall.

Y. Weiss and W. T. Freeman 2000. Correctness of belief propagation in Gaussian graphical models of arbitrary topology. In S.A. Solla, T. K. Leen, and K.-R. Müller (eds) *Advances in Neural Information Processing Systems 12*, MIT Press.

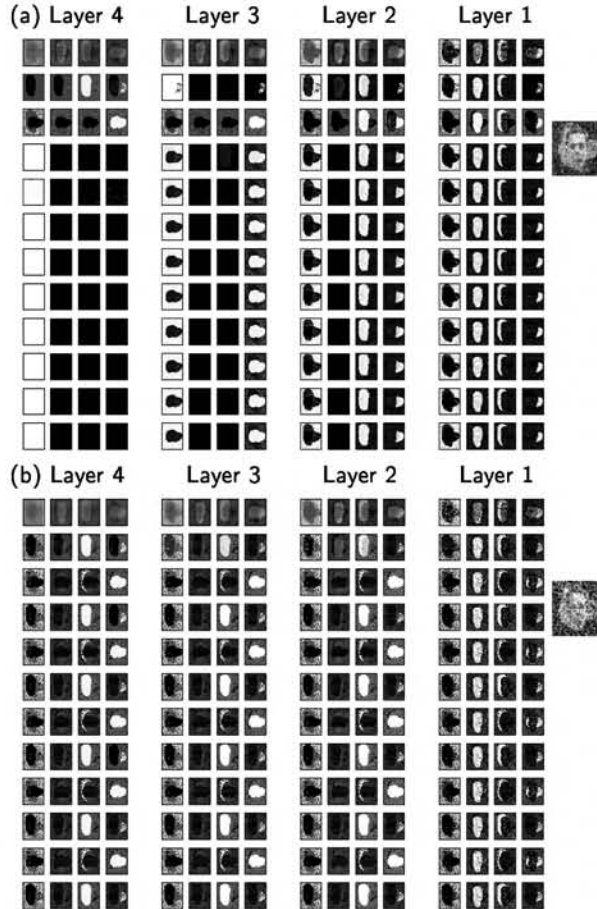

Figure 7: (a) Probabilistic segmentation maps for each layer (column) after each iteration (row) of probability propagation for the image on the far right. (b) When a large amount of noise is added to the image, the network oscillates between interpretations.